# Online Clustering of Moving Hyperplanes

**René Vidal**

Center for Imaging Science, Department of Biomedical Engineering, Johns Hopkins University
308B Clark Hall, 3400 N. Charles St., Baltimore, MD 21218, USA
rvidal@cis.jhu.edu

## Abstract

We propose a recursive algorithm for clustering trajectories lying in multiple moving hyperplanes. Starting from a given or random initial condition, we use normalized gradient descent to update the coefficients of a time varying polynomial whose degree is the number of hyperplanes and whose derivatives at a trajectory give an estimate of the vector normal to the hyperplane containing that trajectory. As time proceeds, the estimates of the hyperplane normals are shown to track their true values in a stable fashion. The segmentation of the trajectories is then obtained by clustering their associated normal vectors. The final result is a simple recursive algorithm for segmenting a variable number of moving hyperplanes. We test our algorithm on the segmentation of dynamic scenes containing rigid motions and dynamic textures, e.g., a bird floating on water. Our method not only segments the bird motion from the surrounding water motion, but also determines patterns of motion in the scene (e.g., periodic motion) directly from the temporal evolution of the estimated polynomial coefficients. Our experiments also show that our method can deal with appearing and disappearing motions in the scene.

## 1    Introduction

Principal Component Analysis (PCA) [1] refers to the problem of fitting a linear subspace $S \subset \mathbb{R}^D$ of unknown dimension $d < D$ to $N$ sample points $\boldsymbol{X} = \{\boldsymbol{x}_i \in S\}_{i=1}^N$. A natural extension of PCA is subspace clustering, which refers to the problem of fitting a union of $n \geq 1$ linear subspaces $\{S_j \subset \mathbb{R}^D\}_{j=1}^n$ of unknown dimensions $d_j = \dim(S_j)$, $0 < d_j < D$, to $N$ points $\boldsymbol{X} = \{\boldsymbol{x}_i \in \mathbb{R}^D\}_{i=1}^N$ drawn from $\cup_{j=1}^n S_j$, without knowing which points belong to which subspace. This problem shows up in a variety of applications in computer vision (image compression, motion segmentation, dynamic texture segmentation) and also in control (hybrid system identification).

Subspace clustering has been an active topic of research over the past few years. Existing methods randomly choose a basis for each subspace, and then iterate between data segmentation and standard PCA. This can be done using methods such as Ksubspaces [2], an extension of Kmeans to the case of subspaces, or Expectation Maximization for Mixtures of Probabilistic PCAs [3]. An alternative algebraic approach, which does not require any initialization, is Generalized PCA (GPCA) [4]. In GPCA the data points are first projected onto a low-dimensional subspace. Then, a set of polynomials is fitted to the projected data points and a basis for each one of the projected subspaces is obtained from the derivatives of these polynomials at the data points.

Unfortunately, all existing subspace clustering methods are *batch*, i.e. the subspace bases and the segmentation of the data are obtained after all the data points have been collected. In addition, existing methods are designed for clustering data lying in a collection of *static subspaces*, i.e. the subspace bases do not change as a function of time. Therefore, when these methods are applied to time-series data, e.g., dynamic texture segmentation, one typically applies them to a moving time window, under the assumption that the subspaces are static within that window. A major disadvantage of this approach is that it does not incorporate temporal coherence, because the segmentation

and the bases at time $t + 1$ are obtained independently from those at time $t$. Also, this approach is computationally expensive, since a new subspace clustering problem is solved at each time instant.

In this paper, we propose a computationally simple and temporally coherent *online* algorithm for clustering point trajectories lying in a variable number of *moving hyperplanes*. We model a union of $n$ moving hyperplanes in $\mathbb{R}^D$, $S_j(t) = \{\boldsymbol{x} \in \mathbb{R}^D : \boldsymbol{b}_j^\top(t)\boldsymbol{x} = \boldsymbol{0}\}$, $j = 1, \ldots, n$, where $\boldsymbol{b}(t) \in \mathbb{R}^D$, as the zero set of a polynomial with time varying coefficients. Starting from an initial polynomial at time $t$, we compute an update of the polynomial coefficients using normalized gradient descent. The hyperplane normals are then estimated from the derivatives of the new polynomial at each trajectory. The segmentation of the trajectories is obtained by clustering their associated normal vectors. As time proceeds, new data are added, and the estimates of the polynomial coefficients are more accurate, because they are based on more observations. This not only makes the segmentation of the data more accurate, but also allows us to handle a variable number of hyperplanes. We test our approach on the challenging problem of segmenting dynamic textures from rigid motions in video.

## 2 Recursive estimation of a single hyperplane

In this section, we review the normalized gradient algorithm for estimating a single hyperplane. We consider both static and moving hyperplanes, and analyze the stability of the algorithm in each case.

**Recursive linear regression.** For the sake of simplicity, let us first revisit a simple linear regression problem in which we are given measurements $\{\boldsymbol{x}(t), y(t)\}$ related by the equation $y(t) = \boldsymbol{b}^\top \boldsymbol{x}(t)$. At time $t$, we seek an estimate $\hat{\boldsymbol{b}}(t)$ of $\boldsymbol{b}$ that minimizes $f(\boldsymbol{b}) = \sum_{\tau=1}^{t}(y(\tau) - \boldsymbol{b}^\top \boldsymbol{x}(\tau))^2$. A simple strategy is to recursively update $\hat{\boldsymbol{b}}(t)$ by following the negative of the gradient direction at time $t$,

$$\boldsymbol{v}(t) = -(\hat{\boldsymbol{b}}(t)^\top \boldsymbol{x}(t) - y(t))\boldsymbol{x}(t). \tag{1}$$

However, it is better to normalize this gradient in order to achieve better convergence properties. As shown in Theorem 2.8, page 77 of [5], the following *normalized gradient recursive identifier*

$$\hat{\boldsymbol{b}}(t + 1) = \hat{\boldsymbol{b}}(t) - \mu \frac{(\hat{\boldsymbol{b}}(t)^\top \boldsymbol{x}(t) - y(t))}{1 + \mu \|\boldsymbol{x}(t)\|^2} \boldsymbol{x}(t), \tag{2}$$

where $\mu > 0$ is a fixed parameter, is such that $\hat{\boldsymbol{b}}(t) \to \boldsymbol{b}$ exponentially if the regressors $\{\boldsymbol{x}(t)\}$ are *persistently exciting*, i.e. if there is an $S \in \mathbb{N}$ and $\rho_1, \rho_2 > 0$ such that for all $m$

$$\rho_1 I_D \prec \sum_{t=m}^{m+S} \boldsymbol{x}(t)\boldsymbol{x}(t)^\top \prec \rho_2 I_D, \tag{3}$$

where $A \prec B$ means that $(B - A)$ is positive definite and $I_D$ is the identity matrix in $\mathbb{R}^D$. Intuitively, the condition on the left hand side of (3) means that the data has to be persistently "rich enough" in time in order to uniquely estimate the vector $\boldsymbol{b}$, while the condition on the right hand side is needed for stability purposes, as it imposes a uniform upper bound on the covariance of the data.

Consider now a modification of the linear regression problem in which the parameter vector varies with time, i.e. $y(t) = \boldsymbol{b}^\top(t)\boldsymbol{x}(t)$. As shown in [6], if the regressors $\{\boldsymbol{x}(t)\}$ are persistently exciting and the sequence $\{\boldsymbol{b}(t+1) - \boldsymbol{b}(t)\}$ is $L_2$-stable, i.e. $\sup_{t \geq 1} \|\boldsymbol{b}(t+1) - \boldsymbol{b}(t)\|^2 < \infty$, then the normalized gradient recursive identifier (2) produces an estimate $\hat{\boldsymbol{b}}(t)$ of $\boldsymbol{b}(t)$ such that $\{\boldsymbol{b}(t) - \hat{\boldsymbol{b}}(t)\}$ is $L_2$-stable.

**Recursive hyperplane estimation.** Let $\{\boldsymbol{x}(t)\}$ be a set of measurements lying in the moving hyperplane $S(t) = \{\boldsymbol{x} \in \mathbb{R}^D : \boldsymbol{b}^\top(t)\boldsymbol{x} = 0\}$. At time $t$, we seek an estimate $\hat{\boldsymbol{b}}(t)$ of $\boldsymbol{b}(t)$ that minimizes the error $f(\boldsymbol{b}(t)) = \sum_{\tau=1}^{t}(\boldsymbol{b}^\top(\tau)\boldsymbol{x}(\tau))^2$ subject to the constraint $\|\boldsymbol{b}(t)\| = 1$. Notice that the main difference between linear regression and hyperplane estimation is that in the latter case the parameter vector $\boldsymbol{b}(t)$ is constrained to lie in the unit sphere $\mathbb{S}^{D-1}$. Therefore, instead of applying standard gradient descent as in (2), we must follow the negative gradient direction along the geodesic curve in $\mathbb{S}^{D-1}$ passing through $\hat{\boldsymbol{b}}(t)$. As shown in [7], the geodesic curve passing through $\boldsymbol{b} \in \mathbb{S}^{D-1}$ along the tangent vector $\boldsymbol{v} \in T\mathbb{S}^{D-1}$ is $\boldsymbol{b}\cos(\|\boldsymbol{v}\|) + \frac{\boldsymbol{v}}{\|\boldsymbol{v}\|}\sin(\|\boldsymbol{v}\|)$. Therefore, the update equation for the normalized gradient recursive identifier on the sphere is

$$\hat{\boldsymbol{b}}(t + 1) = \hat{\boldsymbol{b}}(t)\cos(\|\boldsymbol{v}(t)\|) + \frac{\boldsymbol{v}(t)}{\|\boldsymbol{v}(t)\|}\sin(\|\boldsymbol{v}(t)\|), \tag{4}$$

where the negative normalized gradient is computed as

$$\boldsymbol{v}(t) = -\mu\big(I_D - \hat{\boldsymbol{b}}(t)\hat{\boldsymbol{b}}^\top(t)\big)\frac{(\hat{\boldsymbol{b}}^\top(t)\boldsymbol{x}(t))\boldsymbol{x}(t)}{1+\mu\|\boldsymbol{x}(t)\|^2}. \tag{5}$$

Notice that the gradient on the sphere is essentially the same as the Euclidean gradient, except that it needs to be projected onto the subspace orthogonal to $\hat{\boldsymbol{b}}(t)$ by the matrix $I_D - \hat{\boldsymbol{b}}(t)\hat{\boldsymbol{b}}^\top(t) \in \mathbb{R}^{D\times(D-1)}$. Another difference between recursive linear regression and recursive hyperplane estimation is that the persistence of excitation condition (3) needs to be modified to

$$\rho_1 I_{D-1} \prec \sum_{t=m}^{m+S} P_{\boldsymbol{b}(t)}\boldsymbol{x}(t)\boldsymbol{x}(t)^\top P_{\boldsymbol{b}(t)}^\top \prec \rho_2 I_{D-1}, \tag{6}$$

where the projection matrix $P_{\boldsymbol{b}(t)} \in \mathbb{R}^{(D-1)\times D}$ onto the orthogonal complement of $\boldsymbol{b}(t)$ accounts for the fact that $\|\boldsymbol{b}(t)\| = 1$. Under persistence of excitation condition (6), if $\boldsymbol{b}(t) = \boldsymbol{b}$ the identifier (4) is such that $\hat{\boldsymbol{b}}(t) \to \boldsymbol{b}$ exponentially, while if $\{\boldsymbol{b}(t+1) - \boldsymbol{b}(t)\}$ is $L_2$-stable, so is $\{\boldsymbol{b}(t) - \hat{\boldsymbol{b}}(t)\}$.

## 3 Recursive segmentation of a known number of moving hyperplanes

In this section, we generalize the recursive identifier (4) and its stability properties to the case of $N$ trajectories $\{\boldsymbol{x}_i(t)\}_{i=1}^N$ lying in $n$ hyperplanes $\{S_j(t)\}_{j=1}^n$. In principle, we could apply the identifier (2) to each one of the hyperplanes. However, as we do not know the segmentation of the data, we do not know which data to use to update each one of the $n$ identifiers. In the approach, the $n$ hyperplanes are represented with a single polynomial whose coefficients do not depend on the segmentation of the data. By updating the coefficients of this polynomial, we can simultaneously estimate all the hyperplanes, without first clustering the point trajectories.

**Representing moving hyperplanes with a time varying polynomial.** Let $\boldsymbol{x}(t)$ be an arbitrary point in one of the $n$ hyperplanes. Then there is a vector $\boldsymbol{b}_j(t)$ normal to $S_j(t)$ such that $\boldsymbol{b}_j^\top(t)\boldsymbol{x}(t) = 0$. Thus, the following homogeneous polynomial of degree $n$ in $D$ variables must vanish at $\boldsymbol{x}(t)$:

$$p_n(\boldsymbol{x}(t), t) = \big(\boldsymbol{b}_1^\top(t)\boldsymbol{x}(t)\big)\big(\boldsymbol{b}_2^\top(t)\boldsymbol{x}(t)\big)\cdots\big(\boldsymbol{b}_n^\top(t)\boldsymbol{x}(t)\big) = 0. \tag{7}$$

This homogeneous polynomial can be written as a linear combination of all the monomials of degree $n$ in $\boldsymbol{x}$, $\boldsymbol{x}^I = x_1^{n_1}x_2^{n_2}\cdots x_D^{n_D}$ with $0 \le n_k \le n$ for $k = 1,\ldots,D$, and $n_1 + n_2 + \cdots + n_D = n$, as

$$p_n(\boldsymbol{x}, t) \doteq \sum c_{n_1,\ldots,n_D}(t)x_1^{n_1}\cdots x_D^{n_D} = \boldsymbol{c}(t)^\top \nu_n(\boldsymbol{x}) = 0, \tag{8}$$

where $c_I(t) \in \mathbb{R}$ represents the coefficient of the monomial $\boldsymbol{x}^I$. The map $\nu_n : \mathbb{R}^D \to \mathbb{R}^{M_n(D)}$ is known as the *Veronese map* of degree $n$, which is defined as [8]:

$$\nu_n : [x_1,\ldots,x_D]^\top \mapsto [\ldots, \boldsymbol{x}^I, \ldots]^\top, \tag{9}$$

where $I$ is chosen in the degree-lexicographic order and $M_n(D) = \binom{n+D-1}{n}$ is the total number of independent monomials. Notice that since the normal vectors $\{\boldsymbol{b}_j(t)\}$ are time dependent, the vector of coefficients $\boldsymbol{c}(t)$ is also time dependent. Since both the normal vectors and the coefficient vector are defined up to scale, we will assume that $\|\boldsymbol{b}_j(t)\| = \|\boldsymbol{c}(t)\| = 1$, without loss of generality.

**Recursive identification of the polynomial coefficients.** Thanks to the polynomial equation (8), we now propose a new online hyperplane clustering algorithm that operates on the polynomial coefficients $\boldsymbol{c}(t)$, rather than on the normal vectors $\{\boldsymbol{b}_j(t)\}_{i=1}^n$. The advantage of doing so is that $\boldsymbol{c}(t)$ does not depend on which hyperplane the measurement $\boldsymbol{x}(t)$ belongs to. Our method operates as follows. At each time $t$, we seek to find an estimate $\hat{\boldsymbol{c}}(t)$ of $\boldsymbol{c}(t)$ that minimizes

$$f(\boldsymbol{c}(t)) = \frac{1}{N}\sum_{\tau=1}^t \sum_{i=1}^N (\boldsymbol{c}(\tau)^\top \nu_n(\boldsymbol{x}_i(\tau)))^2. \tag{10}$$

By using normalized gradient descent on $\mathbb{S}^{M_n(D)-1}$, we obtain the following recursive identifier

$$\hat{\boldsymbol{c}}(t+1) = \hat{\boldsymbol{c}}(t)\cos(\|\boldsymbol{v}(t)\|) + \frac{\boldsymbol{v}(t)}{\|\boldsymbol{v}(t)\|}\sin(\|\boldsymbol{v}(t)\|), \tag{11}$$

where the negative normalized gradient is computed as

$$\boldsymbol{v}(t) = -\mu\big(I_{M_n(D)} - \hat{\boldsymbol{c}}(t)\hat{\boldsymbol{c}}^{\top}(t)\big)\frac{\sum_{i=1}^{N}(\hat{\boldsymbol{c}}^{\top}(t)\nu_n(\boldsymbol{x}_i(t)))\nu_n(\boldsymbol{x}_i(t))/N}{1 + \mu\sum_{i=1}^{N}\|\nu_n(\boldsymbol{x}_i(t))\|^2/N}. \tag{12}$$

Notice that (11) reduces to (4) and (12) reduces to (5) if $n = 1$ and $N = 1$.

**Recursive identification of the hyperplane normals.** Given an estimate of $\boldsymbol{c}(t)$, we may obtain an estimate of the vector normal to the hyperplane containing a trajectory $\boldsymbol{x}(t)$ from the derivative of the polynomial $\hat{p}_n(\boldsymbol{x}, t) = \hat{\boldsymbol{c}}^{\top}(t)\nu_n(\boldsymbol{x})$ at $\boldsymbol{x}(t)$ as

$$\hat{\boldsymbol{b}}(\boldsymbol{x}(t)) = \frac{D\nu_n^{\top}(\boldsymbol{x}(t))\hat{\boldsymbol{c}}(t)}{\|D\nu_n^{\top}(\boldsymbol{x}(t))\hat{\boldsymbol{c}}(t)\|}, \tag{13}$$

where $D\nu_n(\boldsymbol{x})$ is the Jacobian of $\nu_n$ at $\boldsymbol{x}$. We choose the derivative of $\hat{p}_n$ to estimate the normal vector $\boldsymbol{b}_j(t)$, because if $\boldsymbol{x}(t)$ is a trajectory in the $j$th hyperplane, then $\boldsymbol{b}_j^{\top}(t)\boldsymbol{x}(t) = 0$, hence the derivative of the true polynomial $p_n$ at the trajectory gives

$$Dp_n(\boldsymbol{x}(t), t) = \frac{\partial p_n(\boldsymbol{x}(t), t)}{\partial \boldsymbol{x}(t)} = \sum_{k=1}^{n}\prod_{\ell \neq k}(\boldsymbol{b}_\ell^{\top}(t)\boldsymbol{x}(t))\boldsymbol{b}_k(t) \sim \boldsymbol{b}_j(t). \tag{14}$$

**Stability of the recursive identifier.** Since in practice we do not know the true polynomial coefficients $\boldsymbol{c}(t)$, and we estimate $\boldsymbol{b}(t)$ from $\hat{\boldsymbol{c}}(t)$, we need to show that both $\hat{\boldsymbol{c}}(t)$ and $\hat{\boldsymbol{b}}(\boldsymbol{x}(t))$ track their true values in a stable fashion. Theorem 1 shows that this is the case. Notice that the persistence of excitation condition for multiple hyperplanes (15) is essentially the same as the one for a single hyperplane (6), but properly modified to take into account that the regressors are a set of trajectories in the embedded space $\{\nu_n(\boldsymbol{x}_i(t))\}_{i=1}^{N}$, rather than a single trajectory in the original space $\{\boldsymbol{x}(t)\}$.

**Theorem 1** *Let $P_{\boldsymbol{c}(t)} \in \mathbb{R}^{(M_n(D)-1)\times M_n(D)}$ be a projection matrix onto the orthogonal complement of $\boldsymbol{c}(t)$. Consider the recursive identifier (11)–(13) and assume that the embedded regressors $\{\nu_n(\boldsymbol{x}_i(t))\}_{i=1}^{N}$ are persistently exciting, i.e. there exist $\rho_1, \rho_2 > 0$ and $S \in \mathbb{N}$ such that for all $m$*

$$\rho_1 I_{M_n(D)-1} \prec \sum_{t=m}^{m+S}\sum_{i=1}^{N}P_{\boldsymbol{c}(t)}\nu_n(\boldsymbol{x}_i(t))\nu_n^{\top}(\boldsymbol{x}_i(t))P_{\boldsymbol{c}(t)}^{\top} \prec \rho_2 I_{M_n(D)-1}. \tag{15}$$

*Then the sequence $\boldsymbol{c}(t) - \hat{\boldsymbol{c}}(t)$ is $L_2$-stable. Furthermore, if a trajectory $\boldsymbol{x}(t)$ belongs to the $j$th hyperplane, then the corresponding $\hat{\boldsymbol{b}}(\boldsymbol{x}(t))$ in (13) is such that $\boldsymbol{b}_j(t) - \hat{\boldsymbol{b}}(\boldsymbol{x}(t))$ is $L_2$-stable. If in addition the hyperplanes are static, then $\boldsymbol{c}(t) - \hat{\boldsymbol{c}}(t) \to \boldsymbol{0}$ and $\boldsymbol{b}_j(t) - \hat{\boldsymbol{b}}(\boldsymbol{x}(t)) \to \boldsymbol{0}$ exponentially.*

*Proof.* [Sketch only] When the hyperplanes are static, the exponential convergence of $\boldsymbol{c}(t)$ to $\boldsymbol{c}$ follows with minor modifications from Theorem 2.8, page 77 of [5]. This implies that $\exists \kappa, \lambda > 0$ such that $\|\hat{\boldsymbol{c}}(t) - \boldsymbol{c}\| < \kappa\lambda^{-t}$. Also, since the vectors $\boldsymbol{b}_1, \ldots, \boldsymbol{b}_n$ are different, the polynomial $\boldsymbol{c}^{\top}\nu_n(\boldsymbol{x})$ has no repeated factor. Therefore, there is a $\delta > 0$ and a $T > 0$ such that for all $t > T$ we have $\|D\nu_n(\boldsymbol{x}(t))^{\top}\boldsymbol{c}\| \geq \delta$ and $\|D\nu_n(\boldsymbol{x}(t))^{\top}\hat{\boldsymbol{c}}(t)\| \geq \delta$ (see proof of Theorem 3 in [9] for the latter claim). Combining this with $\|\hat{\boldsymbol{c}}\| \leq \|\boldsymbol{c}\| + \|\hat{\boldsymbol{c}} - \boldsymbol{c}\|$ and $\|\boldsymbol{c}\| = 1$, we obtain that when $\boldsymbol{x}(t) \in S_j$,

$$\|\boldsymbol{b}_j - \hat{\boldsymbol{b}}(\boldsymbol{x}(t))\| = \left\|\frac{\|D\nu_n^{\top}(\boldsymbol{x}(t))\hat{\boldsymbol{c}}(t)\|D\nu_n^{\top}(\boldsymbol{x}(t))\boldsymbol{c} - \|D\nu_n^{\top}(\boldsymbol{x}(t))\boldsymbol{c}\|D\nu_n^{\top}(\boldsymbol{x}(t))\hat{\boldsymbol{c}}(t)}{\|D\nu_n^{\top}(\boldsymbol{x}(t))\hat{\boldsymbol{c}}(t)\|\|D\nu_n^{\top}(\boldsymbol{x}_t)\boldsymbol{c}\|}\right\|$$

$$\leq \frac{\left\|\|(D\nu_n^{\top}(\boldsymbol{x}(t))(\hat{\boldsymbol{c}}(t) - \boldsymbol{c})\|D\nu_n^{\top}(\boldsymbol{x}(t))\boldsymbol{c} - \|D\nu_n^{\top}(\boldsymbol{x}(t))\boldsymbol{c}\|D\nu_n^{\top}(\boldsymbol{x}(t))(\hat{\boldsymbol{c}}(t) - \boldsymbol{c}))\right\|}{\delta^2}$$

$$\leq 2\frac{\|D\nu_n^{\top}(\boldsymbol{x}(t))(\hat{\boldsymbol{c}}(t) - \boldsymbol{c})\|\|D\nu_n^{\top}(\boldsymbol{x}(t))\boldsymbol{c}\|}{\delta^2} \leq 2\frac{\|D\nu_n^{\top}(\boldsymbol{x}(t))\|^2\|\hat{\boldsymbol{c}}(t) - \boldsymbol{c})\|}{\delta^2} = 2\frac{\alpha_n^2 E_n^2\kappa\lambda^{-t}}{\delta^2},$$

showing that $\hat{\boldsymbol{b}}(\boldsymbol{x}(t)) \to \boldsymbol{b}_j$ exponentially. In the last step we used the fact that for all $\boldsymbol{x} \in \mathbb{R}^D$ there is a constant matrix of exponents $E_{kn} \in \mathbb{R}^{M_n(D)\times M_{n-1}(D)}$ such that $\partial\nu_n(\boldsymbol{x})/\partial x_k = E_{kn}\nu_{n-1}(\boldsymbol{x})$. Therefore, $\|D\nu_n(\boldsymbol{x})\| \leq E_n\|\nu_{n-1}(\boldsymbol{x})\| = E_n\sqrt[\frac{n}{n-1}]{\|\nu_n(\boldsymbol{x})\|} \leq \alpha_n E_n$, where $E_n = \max(\|E_{kn}\|)$ and $\alpha_n = \sqrt[2(n-1)]{\rho_2}$. Consider now the case in which the hyperplanes are moving. Since $\mathbb{S}^{D-1}$ is compact, the sequences $\{\boldsymbol{b}_j(t+1) - \boldsymbol{b}_j(t)\}_{j=1}^{n}$ are trivially $L_2$-stable, hence so is the sequence $\{\boldsymbol{c}(t+1) - \boldsymbol{c}(t)\}$. The $L_2$-stability of $\{\boldsymbol{c}(t) - \hat{\boldsymbol{c}}(t)\}$ and $\{\boldsymbol{b}_j(t) - \hat{\boldsymbol{b}}(t)\}$ follows. ∎

**Segmentation of the point trajectories.** Theorem 1 provides us with a method for computing an estimate $\hat{\boldsymbol{b}}(\boldsymbol{x}_i(t))$ for the normal to the hyperplane passing through each one of the $N$ trajectories $\{\boldsymbol{x}_i(t) \in \mathbb{R}^D\}_{i=1}^N$ at each time instant. The next step is to cluster these normals into $n$ groups, thereby segmenting the $N$ trajectories. We do so by using a recursive version of the K-means algorithm, adapted to vectors on the unit sphere. Essentially, at each $t$, we seek the normal vectors $\hat{\boldsymbol{b}}_j(t) \in \mathbb{S}^{D-1}$ and the membership of $w_{ij}(t) \in \{0,1\}$ of trajectory $i$ to hyperplane $j$ that maximize

$$f(\{w_{ij}(t)\}, \{\hat{\boldsymbol{b}}_j(t)\}) = \sum_{i=1}^N \sum_{j=1}^n w_{ij}(t)(\hat{\boldsymbol{b}}_j^\top(t)\hat{\boldsymbol{b}}(\boldsymbol{x}_i(t)))^2. \tag{16}$$

The main difference with K-means is that we maximize the dot product of each data point with the cluster center, rather than minimizing the distance. Therefore, the cluster center is given by the principal component of each group, rather than the mean. In order to obtain temporally coherent estimates of the normal vectors, we use the estimates at time $t$ to initialize the iterations at time $t+1$.

---

**Algorithm 1 (Recursive hyperplane segmentation)**

---

**Initialization step**

1: Randomly choose $\{\hat{\boldsymbol{b}}_j(1)\}_{j=1}^n$ and $\hat{c}(1)$, or else apply the GPCA algorithm to $\{\boldsymbol{x}_i(1)\}_{i=1}^N$.

**For each $t \geq 1$**

1: Update the coefficients of the polynomial $\hat{p}_n(\boldsymbol{x}(t), t) = \hat{c}(t)^\top \nu_n(\boldsymbol{x}(t))$ using the recursive procedure

$$\hat{c}(t+1) = \hat{c}(t) \cos(\|\boldsymbol{v}(t)\|) + \frac{\boldsymbol{v}(t)}{\|\boldsymbol{v}(t)\|} \sin(\|\boldsymbol{v}(t)\|),$$

$$\boldsymbol{v}(t) = -\mu\big(I_{M_n(D)} - \hat{c}(t)\hat{c}^\top(t)\big)\frac{\sum_{i=1}^N (\hat{c}^\top(t)\nu_n(\boldsymbol{x}_i(t)))\nu_n(\boldsymbol{x}_i(t))/N}{1 + \mu \sum_{i=1}^N \|\nu_n(\boldsymbol{x}_i(t))\|^2/N}.$$

2: Solve for the normal vectors from the derivatives of $\hat{p}_n$ at the given trajectories

$$\hat{\boldsymbol{b}}(\boldsymbol{x}_i(t)) = \frac{D\nu_n^\top(\boldsymbol{x}_i(t))\hat{c}(t)}{\|D\nu_n^\top(\boldsymbol{x}_i(t))\hat{c}(t)\|} \qquad i = 1, \ldots, N.$$

3: Segment the normal vectors using the K-means algorithm on the sphere

   (a) Set $w_{ij}(t) = \begin{cases} 1 & \text{if } j = \arg \max\limits_{k=1,\ldots,n} (\hat{\boldsymbol{b}}_k^\top(t)\hat{\boldsymbol{b}}(\boldsymbol{x}_i(t)))^2 \\ 0 & \text{otherwise} \end{cases}, \qquad i = 1, \ldots, N, j = 1, \ldots, n$

   (b) Set $\hat{\boldsymbol{b}}_j(t) = PCA([w_{1j}(t)\hat{\boldsymbol{b}}(\boldsymbol{x}_1(t)) \quad w_{2j}(t)\hat{\boldsymbol{b}}(\boldsymbol{x}_2(t)) \quad \cdots \quad w_{Nj}(t)\hat{\boldsymbol{b}}(\boldsymbol{x}_N(t))]), \quad j = 1, \ldots, n$

   (c) Iterate (a) and (b) until convergence of $w_{ij}(t)$, and then set $\hat{\boldsymbol{b}}_j(t+1) = \hat{\boldsymbol{b}}_j(t)$.

---

## 4    Recursive segmentation of a variable number of moving hyperplanes

In the previous section, we proposed a recursive algorithm for segmenting $n$ moving hyperplanes under the assumption that $n$ is *known* and *constant* in time. However, in many practical situations the number of hyperplanes may be unknown and time varying. For example, the number of moving objects in a video sequence may change due to objects entering or leaving the camera field of view.

In this section, we consider the problem of segmenting a variable number of moving hyperplanes. We denote by $n(t) \in \mathbb{N}$ the number of hyperplanes at time $t$ and assume we are given an upper bound $n \geq n(t)$. We show that if we apply Algorithm 1 with the number of hyperplanes set to $n$, then we can still recover the correct segmentation of the scene, even if $n(t) < n$. To see this, let us have a close look at the persistence of excitation condition in equation (15) of Theorem 1. Since the condition on the right hand side of (15) holds trivially when the regressors $\boldsymbol{x}_i(t)$ are bounded, the only important condition is the one on the left hand side. Notice that the condition on the left hand side implies that the spatial-temporal covariance matrix of the embedded regressors must be of rank $M_n(D) - 1$ in any time window of size $S$ for some integer $S$. Loosely speaking, the embedded regressors must be "rich enough" either in space or in time.

The case in which there is a $\rho_1 > 0$ such that for all $t$

$$n(t) = n \qquad \text{and} \qquad \sum_{i=1}^N P_{\boldsymbol{c}(t)}\nu_n(\boldsymbol{x}_i(t))\nu_n^\top(\boldsymbol{x}_i(t))P_{\boldsymbol{c}(t)}^\top \succ \rho_1 I_{M_n(D)-1} \tag{17}$$

corresponds to the case of data that is rich in space. In this case, at each time instant we draw data from all $n$ hyperplanes and the data is rich enough to estimate all $n$ hyperplanes at each time instant. In fact, condition (17) is the one required by GPCA [4], which in this case can be applied at each time $t$ independently. Notice also that (17) is equivalent to (15) with $S = 1$.

The case in which $n(t) = 1$ and there are $\rho_1 > 0$, $S \in \mathbb{N}$ and $i \in \{1, \dots, N\}$ such that for all $m$

$$\sum_{t=m}^{m+S} \nu_n(\boldsymbol{x}_i(t))\nu_n^\top(\boldsymbol{x}_i(t)) \succ \frac{\rho_1}{N} I_{M_n(D)-1} \tag{18}$$

corresponds to the case of data that is rich in time. In this case, at each time instant we draw data from a single hyperplane. As time proceeds, however, the data must be persistently drawn from at least $n$ hyperplanes in order for (18) to hold. This can be achieved either by having $n$ different static hyperplanes and persistently drawing data from all of them, or by having less than $n$ moving hyperplanes whose motion is rich enough so that (18) holds.

In summary, as long as the embedded regressors satisfy condition (15) for some upper bound $n$ on the number of hyperplanes, the recursive identifier (11)-(13) will still provide $L_2$-stable estimates of the parameters, even if the number of hyperplanes is unknown and variable, and $n(t) < n$ for all $t$.

## 5  Experiments

**Experiments on synthetic data.** We randomly draw $N = 200$ 3D points lying in $n = 2$ planes and apply a time varying rotation to these points for $t = 1, \dots, 1000$ to generate $N$ trajectories $\{\boldsymbol{x}_i(t)\}_{i=1}^N$. Since the true segmentation is known, we compute the vectors $\{\boldsymbol{b}_j(t)\}$ normal to each plane, and use them to generate the vector of coefficients $\boldsymbol{c}(t)$. We run our algorithm on the so-generated data with $n = 2$, $\mu = 1$ and a random initial estimate for the parameters. We compare these estimates with the ground truth using the percentage of misclassified points. We also consider the error of the polynomial coefficients and the normal vectors by computing the angles between the estimated and true values. Figure 1 shows the true and estimated parameters, as well as the estimation errors. Observe that the algorithm takes about 100 seconds for the errors to stabilize within $1.62°$ for the coefficients, $1.62°$ for the normals, and $4\%$ for the segmentation error.

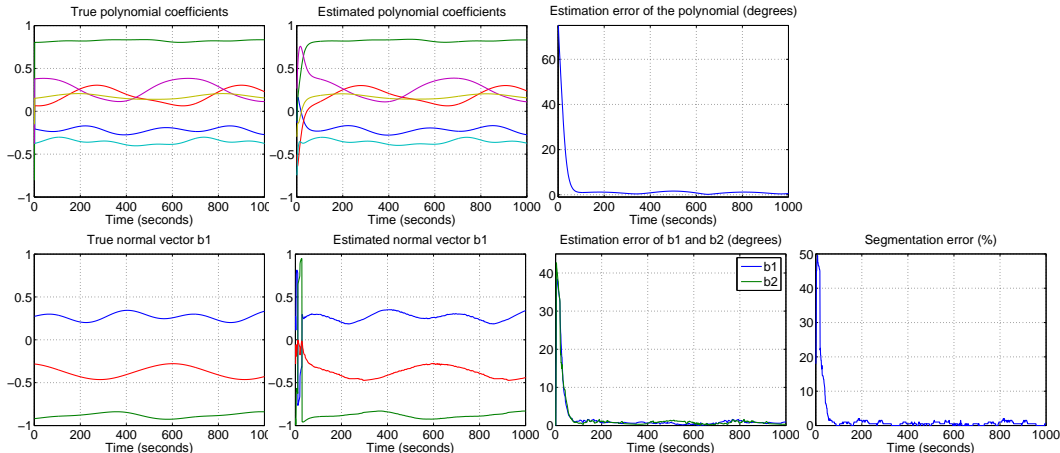

Figure 1: Segmenting 200 points lying on two moving planes in $\mathbb{R}^3$ using our recursive algorithm.

**Segmentation of dynamic textures.** We now apply our algorithm to the problem of segmenting video sequences of dynamic textures, i.e. sequences of nonrigid scenes that exhibit some temporal stationarity, e.g., water, smoke, or foliage. As proposed in [10], one can model the temporal evolution of the image intensities as the output of a linear dynamical system. Since the trajectories of the output of a linear dynamical system live in the so-called observability subspace, the intensity trajectories of pixels associated with a single dynamic texture lie in a subspace. Therefore, the set of all intensity trajectories lie in multiple subspaces, one per dynamic texture.

Given $\gamma$ consecutive frames of a video sequence $\{I(f)\}_{f=t-\gamma+1}^{t}$, we interpret the data as a matrix $W(t) \in \mathbb{R}^{N \times 3\gamma}$, where $N$ is the number of pixels, and 3 corresponds to the three RGB color channels. We obtain a data point $x_i(t) \in \mathbb{R}^D$ from image $I(t)$ by projecting the $i$th row of $W(t)$, $w_i^\top(t)$ onto a subspace of dimension $D$, i.e. $x_i(t) = \Pi w_i(t)$, with $\Pi \in \mathbb{R}^{D \times 3\gamma}$. The projection matrix $\Pi$ can be obtained in a variety of ways. We use the $D$ principal components of the first $\gamma$ frames to define $\Pi$. More specifically, if $W(\gamma) = U\Sigma V^\top$, with $U \in \mathbb{R}^{N \times D}$, $\Sigma \in \mathbb{R}^{D \times D}$ and $V \in \mathbb{R}^{3\gamma \times D}$ is a rank-$D$ approximation of $W(\gamma)$ computed using SVD, then we choose $\Pi = \Sigma^{-1} V^\top$.

We applied our method to a sequence ($110 \times 192, 130$ frames) containing a bird floating on water, while rotating around a fix point. The task is to segment the bird's rigid motion from the water's dynamic texture, while at the same time tracking the motion of the bird. We chose $D = 5$ principal components of the $\gamma = 5$ first frames of the RGB video sequence to project each frame onto a lower dimensional space. Figure 2 shows the segmentation. Although the convergence is not guaranteed with only 130 frames, it is clear that the polynomial coefficients already capture the periodicity of the motion. As shown in the last row of Figure 2, some coefficients of the polynomial oscillate in time. One can notice that the orientation of the bird is related to the value of the coefficient $c_8$. If the bird is facing to the right showing her right side, the value of $c_8$ achieves a local maximum. On the contrary if the bird is oriented to the left, the value of $c_8$ achieves a local minimum. Some irregularities seem to appear at the local minima of this coefficient: they actually correspond to a rapid motion of the bird. One can distinguish three behaviors for the polynomial coefficients: oscillations, pseudo-oscillations or quasi-linearity. For both the oscillations and the pseudo-oscillations the period is identical to the bird's motion period (40 frames). This example shows that the coefficients of the estimated polynomial give useful information about the scene motion.

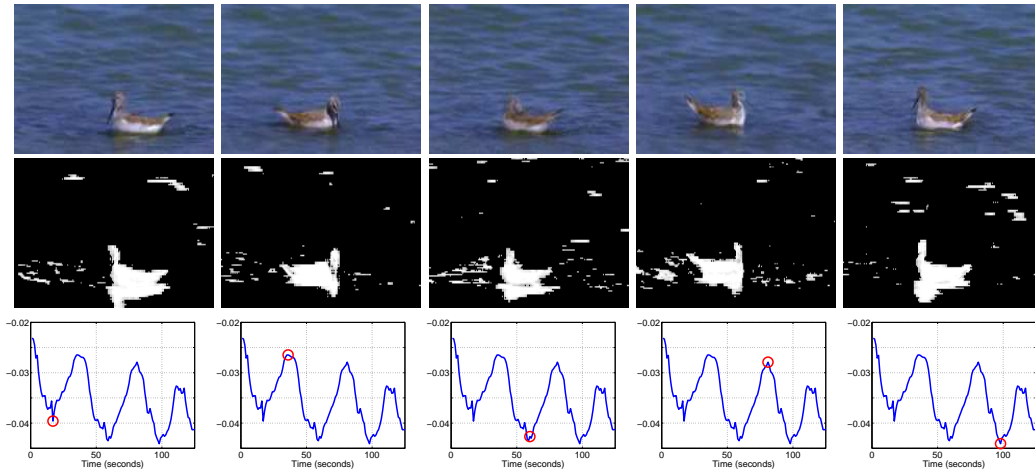

Figure 2: Segmenting a bird floating on water. Top: frames 17, 36, 60, 81, and 98 of the sequence. Middle: segmentation obtained using our method. Bottom: temporal evolution of $c_8$ during the video sequence, with the red dot indicating the location of the corresponding frame in this evolution.

To test the performance of our method on a video sequence with a variable number of motions, we extracted a sub-clip of the bird sequence ($55 \times 192, 130$ frames) in which the camera moves up at 1 pixel/frame until the bird disappears at $t = 51$. The camera stays stationary from $t = 56$ to $t = 66$, and then moves down at 1 pixel/frame, the bird reappears at $t = 76$. We applied both GPCA and our method initialized with GPCA to this video sequence. For GPCA we used a moving window of $\gamma = 5$ frames. For our method we chose $D = 5$ principal components of the $\gamma = 5$ first frames of the RGB video sequence to project each frame onto a fixed lower dimensional space. We set the parameter of the recursive algorithm to $\mu = 1$. Figure 3 shows the segmentation results. Notice that both methods give excellent results during the first few frames, when both the bird and the water are present. This is expected, as our method is initialized with GPCA. Nevertheless, notice that the performance of GPCA deteriorates dramatically when the bird disappears, because GPCA overestimates the number of hyperplanes, whereas our method is robust to this change and keeps segmenting the scene correctly, i.e. assigning all the pixels to the background. When the bird reappears, our method detects the bird correctly from the first frame whereas GPCA produces

a wrong segmentation for the first frames after the bird reappears. Towards the end of the sequence, both algorithms give a good segmentation. This demonstrates that our method has the ability to deal with a variable number of motions, while GPCA has not. In addition the fixed projection and the recursive estimation of the polynomial coefficients make our method much faster than GPCA.

Sequence

GPCA

Our method

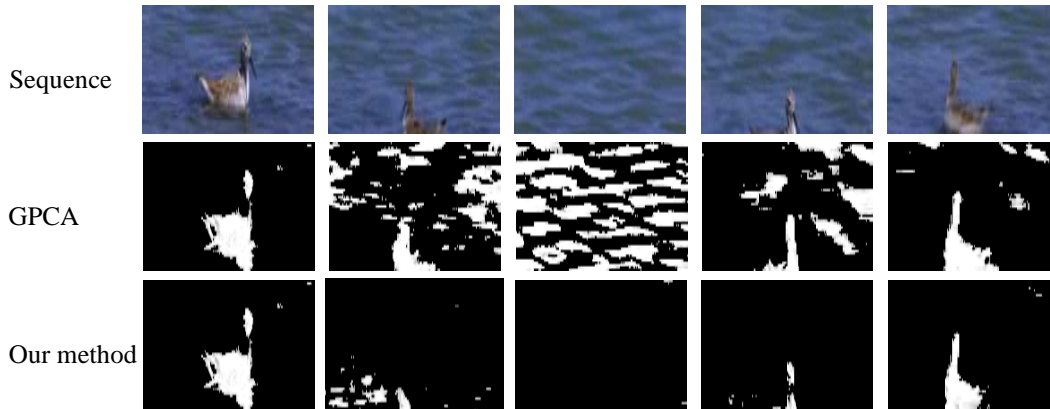

Figure 3: Segmenting a video sequence with a variable number of dynamic textures. Top: frames 1, 24, 65, 77, and 101. Middle: segmentation with GPCA. Bottom: segmentation with our method.

## 6   Conclusions

We have proposed a simple recursive algorithm for segmenting trajectories lying in a variable number of moving hyperplanes. The algorithm updates the coefficients of a polynomial whose derivatives give the normals to the moving hyperplanes as well as the segmentation of the trajectories. We applied our method successfully to the segmentation of videos containing multiple dynamic textures.

## Acknowledgments

The author acknowledges the support of grants NSF CAREER IIS-04-47739, NSF EHS-05-09101 and ONR N00014-05-10836.

## References

[1] I. Jolliffe. *Principal Component Analysis*. Springer-Verlag, New York, 1986.

[2] J. Ho, M.-H. Yang, J. Lim, K.-C. Lee, and D. Kriegman. Clustering apperances of objects under varying illumination conditions. In *IEEE Conference on Computer Vision and Pattern Recognition*, volume 1, pages 11–18, 2003.

[3] M. Tipping and C. Bishop. Mixtures of probabilistic principal component analyzers. *Neural Computation*, 11(2):443–482, 1999.

[4] R. Vidal, Y. Ma, and S. Sastry. Generalized Principal Component Analysis (GPCA). *IEEE Trans. on Pattern Analysis and Machine Intelligence*, 27(12):1–15, 2005.

[5] B.D.O. Anderson, R.R. Bitmead, C.R. Johnson Jr., P.V. Kokotovic, R.L. Ikosut, I.M.Y. Mareels, L. Praly, and B.D. Riedle. *Stability of Adaptive Systems*. MIT Press, 1986.

[6] L. Guo. Stability of recursive stochastic tracking algorithms. In *IEEE Conf. on Decision & Control*, pages 2062–2067, 1993.

[7] A. Edelman, T. Arias, and S. T. Smith. The geometry of algorithms with orthogonality constraints. *SIAM Journal of Matrix Analysis Applications*, 20(2):303–353, 1998.

[8] J. Harris. *Algebraic Geometry: A First Course*. Springer-Verlag, 1992.

[9] R. Vidal and B.D.O. Anderson. Recursive identification of switched ARX hybrid models: Exponential convergence and persistence of excitation. In *IEEE Conf. on Decision & Control*, pages 32–37, 2004.

[10] G. Doretto, A. Chiuso, Y. Wu, and S. Soatto. Dynamic textures. *International Journal of Computer Vision*, 51(2):91–109, 2003.
